# Visual Sequence Learning in Hierarchical Prediction Networks and Primate Visual Cortex

**Jielin Qiu**[1], **Ge Huang**[2], **Tai Sing Lee**[1,2]
[1]Computer Science Department
[2] Neuroscience Institute
Carnegie Mellon University
Pittsburgh, PA 15213
{jielinq,taislee}@andrew.cmu.edu

## Abstract

In this paper we developed a computational hierarchical network model to understand the spatiotemporal sequence learning effects observed in the primate visual cortex. The model is a hierarchical recurrent neural model that learns to predict video sequences using the incoming video signals as teaching signals. The model performs fast feedforward analysis using a deep convolutional neural network with sparse convolution and feedback synthesis using a stack of LSTM modules. The network learns a representational hierarchy by minimizing its prediction errors of the incoming signals at each level of the hierarchy. We found that recurrent feedback in this network lead to the development of semantic cluster of global movement patterns in the population codes of the units at the lower levels of the hierarchy. These representations facilitate the learning of relationship among movement patterns, yielding state-of-the-art performance in long range video sequence predictions on benchmark datasets. Without further tuning, this model automatically exhibits the neurophysiological correlates of visual sequence memories that we observed in the early visual cortex of awake monkeys, suggesting the principle of self-supervised prediction learning might be relevant to understanding the cortical mechanisms of representational learning.

## 1  Introduction

While the hippocampus is known to play a critical role in encoding episodic memories, the storage of these memories might ultimately rest in the sensory areas of the neocortex [1]. Indeed, a number of neurophysiological studies suggest that neurons throughout the hierarchical visual cortex, including those in the early visual areas such as V1 and V2, might be encoding memories of object images [2] and of visual sequences in cell assemblies [3,4,5,6,7]. These memories, together with the generic statistical priors encoded in receptive fields and connectivity of neurons, serve as internal models of the world for predicting incoming visual experiences. However, it is not clear why early visual cortex needs to be involved in the encoding of spatiotemporal memories and what computational roles it might play.

In this paper, we explored a class of computational models based on predictive self-supervised learning for understanding some neurophysiological learning phenomena observed in the early visual cortex. This class of models uses the incoming visual signals as teaching signals to train neural network using backpropagation [8,9,10,11,12,13]. Recently, a number of hierarchical recurrent neural network models based on this principle, notably PredNet [14] and PredRNN++ [15], have been developed for video prediction with state-of-the-art performance. PredNet in particular was inspired by the predictive coding theory in neuroscience [16,17,13,18,19] and is a legitimate cortical model at a functional level. It learns a LSTM (long short-term memory) model at each level to predict the

errors made in an earlier level of the hierarchical network. It has been demonstrated to be effective in explaining the predictive suppression phenomena in the inferotemporal cortex [63]. However, PredNet only builds a hierarchical representation of errors, where the model of a higher layer learns to predict the prediction errors of the lower layer. It does not build a feature hierarchy. Thus, its ability for long-range video prediction is rather limited. PredRNN++ does build a feature hierarchy, but the generation of prediction is based on a auto-encoder-like feedforward network albeit with local recurrent within each layer. It does not explicitly model the recurrent feedback architecture of the cortex. Nor does it claim any neural plausibility.

We propose Hierarchical Prediction Network (HPNet) as an alternative functional model for the visual cortex incorporating additional neural constraints. It draws on good features from both models. It learns a feature hierarchy while using recurrent feedback to provide top-down synthesis of the expectation at each level. In this paper, we will first demonstrate HPNet's effectiveness in video learning and prediction, with performance superior to PredNet and comparable to PredRNN++, which is a state-of-the-art computer vision deep learning models for video prediction. Then we will present novel findings from a neurophysiological experiment that demonstrates the early visual cortex exhibited similar sensitivity to memories of global movement patterns, and that HPNet can automatically account for the neurophysiological observations without further tuning. These findings suggest that predictive self-supervised learning might be relevant principle for learning representational hierarchy in the visual cortex.

## 2    Related works

Our model integrates ideas from predictive coding models [21,17,14] and associative coding or interactive activation models [22,23]. It is therefore also related to the classical ideas of analysis by synthesis [21,64], counter stream model [62] as well as hierarchical spatiotemporal memory model (HTM) [61]. In contrast to the earlier models that use a feedback path which synthesizes the expectation using linear transform, prediction is synthesized in HPNet (as well as PredNet) by an LSTM circuit at each level under feedback gating from higher levels.

Predictive self-supervised learning has long been advocated as a plausible strategy the brain uses to learn internal representations [8]. Recently, thanks to the development of deep learning technology, self-supervised learning in computer vision [24,12,25,11,26,27], and video prediction [9,28,29,30,10,31,32] have become an active area of research. The large variety of models can be roughly grouped into three categories: autoencoders, DCNN, hierarchy of LSTMs, adversary networks [33,14,34,15,35], as well as variational autoencoders[36,37]. The state-of-the-art hierarchical model for video prediction at the writing of this paper is PredRNN++ [15]. Like PredNet and HPNet, PredRNN++ [15] consists of a stack of LSTM modules, but operates in a feedforward auto-encoder architecture to generate the next video frame. It offers state-of-the-art performance for benchmark performance evaluation, with documented comparisons to other approaches.

Both PredNet and HPNet provides recurrent feedback to the early layers of the network that are analogous to early visual areas (V1, V2 and V4). HPNet in particular predicts that higher order semantic information such as global movement pattern information might transform the population codes in the early visual cortex, resulting in sensitivity to memory of global movement patterns in early visual cortical neurons. Neurons in the inferotemporal cortex (IT) of monkeys are known to exhibit sensitivity to memories of predictable familiar image sequences than to novel sequences [52,53,54], and some sensitivity to memories of grating sequences have been reported in V1 [3,4,5,6,7]. In this study, we presented novel neurophysiological findings demonstrating that early visual cortical (V2) neurons in awake monkeys also demonstrate sensitivity to memories of natural movies of large spatial extent in the form of response suppression to familiar or predicted movies, consistent with the behaviors of model neurons in HPNet.

## 3    Hierarchical Prediction Network

### 3.1    Cortical Modules

HPNet is composed of a stack of Cortical Modules (CM). Each CM can be considered as a visual area along the ventral stream of the primate visual system, such as V1, V2, V4 and IT. We used four Cortical Modules in our experiment. The network contains a fast feedforward path, instantiated by a

deep convolutional neural network (DCNN) that learns a representational hierarchy of features of successive complexity, and a feedback path that mediates the synthesis of a prediction via a Long Short Term Memory (LSTM) module at each level. The prediction is compared against the input signal from the feedforward path at that level, and the prediction error is used to modulate the LSTMs at the same level as well as the level above.

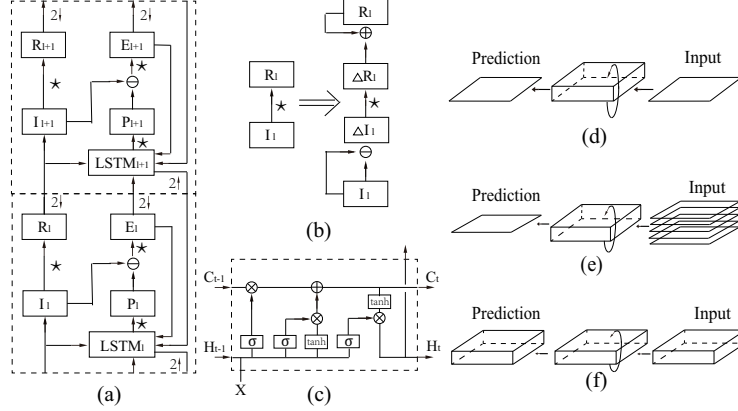

Figure 1: (a) Two successive layers of Cortical Modules in our hierarchical network. The input $I_1$ at the bottom level is a spatiotemporal block of video frames. The $\star$ notation means a convolution along that path. $2\uparrow$ indicates up-sampling or expansion operation. $2\downarrow$ means down-sample or reduction in resolution. $\ominus$ indicates comparator or subtraction operation; (b) The DCNN analysis path is implemented in a sparsified convolution scheme to speed up bottom-up processing; (c) Detailed structure of the standard LSTM used. $C_t$ is the internal state, and $H_t$ is the output. X is external input, which contains multiple sources in our model. (d) Frame-by-frame scheme; (e) Block-by-frame scheme; and (f) Block-by-block scheme, where left and right part indicates output and input respectively with the middle indicating 2D or 3D convolution LSTM.

Figure 1a shows two cascaded CMs. The feedforward path performs convolution (indicated by $\star$) on the input spatiotemporal block $I_l$ with a kernel to produce $R_l$, where $l$ indicates the CM level. $R_l$ is then down-sampled to provide the input $I_{l+1}$ for $CM_{l+1}$ for another round of convolution along the feedforward path. $I_{l+1}$ also goes into $LSTM_{l+1}$ ($LSTM$ in $CM_{l+1}$). In each $CM_l$ level, the bottom-up input $I_l$ is compared with the prediction $P_l$, which is generated from the interpretation output $H_l$ of $LSTM_l$. The prediction error signal is transformed by a convolution into $E_l$, which is fed back to both $LSTM_l$ and $LSTM_{l+1}$ to influence their generation of new interpretations $H_l$ and $H_{l+1}$. To make the timing relationship clear in Algorithm 1, we use $k$ to index a spatiotemporal block in a block sequence, which is extracted from the video input sequence $x_t$ with a stride $s$ that could vary from 1 to $d$, where $d$ is the number of video frames contained in a block. At the bottom input level $I_1^k = (x_{k_s}, .., x_{k_{s+d}})$ $LSTM_l$ at step $k$ integrates the bottom-up feature input $R_{l-1}^k$, the top-down feedback of the higher CM's LSTM's output $H_{l+1}^k$, and the prediction errors $E_{l-1}^k$ and $E_l^{k-1}$ to generate new hypothesis output $H_l^k$, which is then transformed into a new prediction $P_l^k$ (see Algorithm 1 in Supplementary Information (SI)).

## 3.2 Sparse Convolution, Spatiotemporal Blocks and 3D convolutional LSTM

The feedforward DCNN path in Figure 1a runs much faster if the input to each convolution layer is made sparse, as shown in [44]. A scheme has been proposed by [45,46,44] to speed up video processing in convolutional neural net by transmitting the first frame in its entirety, but for the subsequent frames only the frame difference between them is transmitted. Hence, convolution can be performed efficiently on the difference signals $\Delta I_l^k = I_l^k - I_l^{k-1}$ between two consecutive blocks. In the next layer, the resulting $\Delta R_l^k$ is added back to the representation of the last time block $R_l^{k-1}$ to reconstruct the actual higher order feature representation of the input signals $R_l^k$. This allows the network to recover and maintain a full higher order representation $R$ for computation in the next layer while enjoying the benefit of fast computation on sparse data. The same set of sparse kernels was used for processing both the first full frame and the subsequent temporal-difference frames.

Visual neurons' receptive fields are spatiotemporal 3D kernels, rather than 2D. Therefore, we use spatiotemporal block of the input video sequence as input to the neuron in our neural network. The block could slide in time with a temporal stride $s$ as small as one frame or as large as the length of the spatiotemporal block $d$. To process 3D data, a 3D convolutional LSTM is used [20,35]. The details of our 3D convolutional LSTM are specified in the Supplementary Information (SI).

### 3.3 Training and Loss Function

The entire network is trained by minimizing a loss function which is the $L2$ weighted sum of the prediction errors of all the Cortical Modules (CM),

$$L_{loss} = \sum_k \lambda_k \sum_l \frac{\lambda_l}{n_l} \sum_{n_l} (I_l^k - P_l^k)^2 \tag{1}$$

where $k$ indexes the spatiotemporal block sequence, $l$ the CM level, and $n_l$ the number of units in that level; $\lambda_k$ and $\lambda_l$ are weighting factors for time step and CM level, respectively. $I_l^k$ is $k^{th}$ spatiotemporal block input to the CM at level $l$, and $P_l^k$ is the prediction at that level, following the variables' notations above as well as in Figure 1.

$$I_l^k = \begin{cases} MaxPool(ReLU(R_{l-1}^k)) & l > 1 \\ x_t & l = 1 \end{cases} \quad P_l^k = \begin{cases} ReLU(conv(H_l^k)) & l > 1 \\ SATLU(ReLU(conv(H_l^k))) & l = 1 \end{cases} \tag{2}$$

$$H_l^k = 3DconvLSTM(H_l^{k-1}, E_l^{k-1}, MaxPool(ReLU(R_{l-1}^k, E_{l-1}^k)), upsample(H_{l+1}^k)) \tag{3}$$

$$\Delta R_l^k = spconv(I_l^k - I_l^{k-1}), \ E_l^k = spconv(I_l^k - P_l^k) R_l^k = R_l^{k-1} + \Delta R_l^k \tag{4}$$

where $x_t$ is the video input sequence, $H_l^k$ is the output of $LSTM$, $SATLU$ is a saturating non-linearity set at the maximum pixel value ($SATLU(x; p_{max}) := min(p_{max}, x)$, where $spconv$ indicates sparse convolution). The algorithm is shown in Algorithm 1 in the Supplementary Material. For hyperparameter tuning, we adapt PredNet's approach, performing a large grid search in hyperparameter space. We did not try to find the best possible set of parameters, only a set of parameters that beat the state-of-the-art in video prediction. We did not tune our network or other networks in our simulation of the neurophysiological experiments.

## 4 Experimental Results

In this section, to establish computational competency in video prediction, we first evaluate the performance of our model in video prediction using two bench-mark datasets. We will then evaluate the latent representations of the hierarchical network and compare the behaviors of the model units in the network with behaviors of the neurons in the visual cortex in a video sequence learning experiment.

### 4.1 Competency of the Model in Long-Range Video Sequence Prediction

We tested the network with two datasets which were also used as benchmark datasets in PredNet and PredRNN++: (1) synthetic sequences of the Moving-MNIST[1] database and (2) the KTH[2] real world human movement database.

The Moving-MNIST dataset contains synthetic video sequences with two handwritten digits bouncing inside a frame of $64 \times 64$ pixels. Each sequence is 40 frames long and its starting position, hence the speed and direction of the movements, are chosen uniformly at random in [3,4] as [15]. This extraction process is repeated 15000 times, resulting in a training set of 10000 sequences, a validation set of 2000 sequences, and a testing set of 3000 sequences.

The KTH video database [49] contains 2391 real-world sequences of six human actions: walking, jogging, running, boxing, hand waving, and hand clapping, performed by 25 subjects in four different scenarios. We divided video clips across all 6 action categories into a training set of 108717 sequences (persons #1-16) and a test set of 4086 sequences (persons #17-25) as was done in [15], except we

extracted 40-frame sequences. We center-cropped each frame to a 120×120 square and then re-sized it to input frame size of 64×64.

We compared HPNet's video prediction performance, particularly for long-term prediction, with PredNet and PredRNN++. Because these two models work on a frame-to-frame basis to predict the next frame based on all the existing frames, we tested two versions of our network for comparison: (1) Frame-to-Frame (F-F), where we set our spatiotemporal block size of our data to one frame (i.e. $d = 1$) and used 2D convLSTM instead of 3D convLSTM to predict the next frame based on the current and past input frames; (2) Block-to-Block (B-B), our default model using spatiotemporal block as data unit, where the next spatiotemporal block ($d = 5$, $s = 5$) was predicted from the current spatiotemporal block.

We trained all four networks using 40-frame sequences extracted from the two databases in the same way as described in [14,15]. We then compared their performance in predicting the next 20 frames when only the first 20 frames were given. The test sequences were drawn from the same dataset but not in the training set. To predict future frames by dead-reckoning when input was no longer available, PredNet and PredRNN++ simply took the predicted frame and fed into the network as the input frame to generate prediction of the next time step. All models tested have four modules (layers). Both versions of our models and PredNet used the same number of feature channels in each layer, optimized by grid search, i.e. (16,32,64,128) for the Moving-MNIST dataset, and (24,48,96,192) for the KTH dataset. For PredRNN++, we used the same architecture and feature channel numbers provided by [15]. All kernel sizes are either 3×3 (for F-F) or 3×3×3 (for B-B) for all four models. The input image frame's spatial resolution is 64×64. The models were trained and tested on GeForce GTX TITAN X GPUs. We evaluated the prediction performance based on two quantitative index: Mean-Squared Error (MSE) and the Structural Similarity Index Measure (SSIM) [48] of the last 20 frames between the predicted frames and the actual frames. The values of SSIM range from -1 to 1, with larger value indicating greater similarity between the predicted frames and the actual frames.

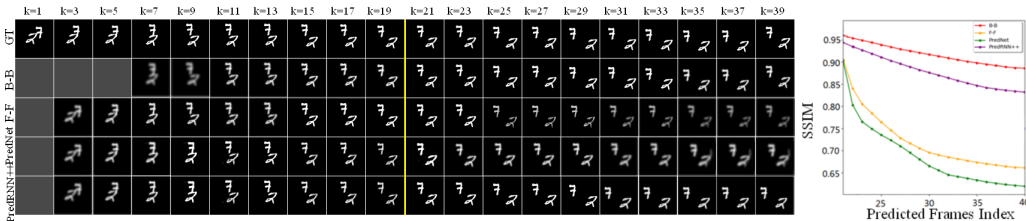

Figure 2: Left panel: Video prediction results on Moving-MNIST dataset, where the first row to last row are ground truth (GT), results from three different version of HPNet (block-to-block (B-B), frame-to-frame (F-F)), PredNet, and PredRNN++, respectively. k=1 to k=19 are predicted frames of the models when the input frames were available. k=21 to k=39 are the "dead-reckoning" predicted frames of the model when there is no input. Right panel: Comparison of the prediction results of the four models for the Moving-MNIST dataset on the last 20 frames in structural similarity measures (SSIM).

Figure 2 and Table 1 show the performance of the four models on the Moving-MNIST dataset. Figure 3 and Table 2 show the performance of the four models on the KTH dataset. In the examples shown in Figure 2 and Figure 3, each test sequence has 40 frames but we show the results every two frames. Actual input was provided only for the first 20 frames, where each frame or block of frames were predicted based on the previous frame or previous block of input. The prediction for the last 20 frames (frame 21 to frame 40) were dead-reckoning prediction. The left panels of both figures show examples of the predicted sequences generated by the four models. The right panels compare the performance of the four models during the last 20 dead-reckoning frames.

For both the synthetic and real world datasets, HPNet using the Block-Block scheme consistently yields the best performance. HPNet in Frame-Frame scheme performs better than PredNet, suggesting that a feature hierarchy is better than a prediction error hierarchy for long-range prediction. However, HPNet in Frame-Frame scheme does not perform as well as PredRNN++ on long range video prediction. The superiority of PredRNN++ in this case is likely because its LSTM at each level is boosted to have access to information from all layers below, rather than from just that layer as in PredNet and HPNet. Hence, it took longer to train but can potentially encode richer movement

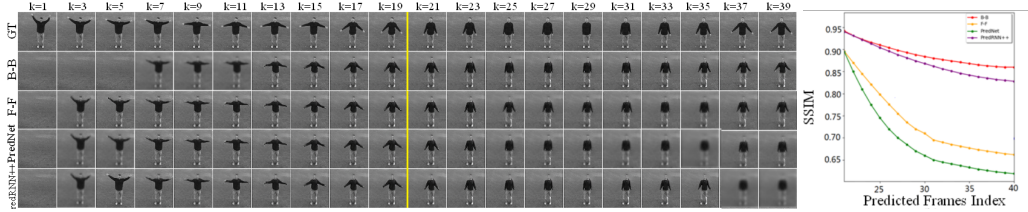

Figure 3: Left panel: Video prediction results on the KTH dataset, where the first row to last row are ground truth (GT), results from block-to-block (B-B), frame-to-frame (F-F), PredNet, and PredRNN++, respectively, same format as Figure 2. Right panel: Comparison of the prediction results of the four models for the KTH dataset on the last 20 frames in structural similarity measures (SSIM).

patterns in its memories for long-term prediction. HPNet might have compensated by using 3D data blocks with 3D convolutional LSTM to achieve better performance. Given that every area in the visual cortex has some recurrent connections to many lower areas (levels) in addition to the adjacent level in the hierarchy, it would be reasonable to incorporate that feature of PredRNN++ to see if further improvement can be obtained.

Table 1: Comparison results of different methods on Moving-MNIST dataset for long time prediction experiment.

| Method | SSIM | MSE |
|---|---|---|
| Ours(B-B) | **0.915** | **65.2** |
| CM+ConvLSTM (F-F) | 0.692 | 89.5 |
| PredNet [14] | 0.658 | 101.2 |
| PredRNN++ [15] | 0.872 | 69.4 |

Table 2: Comparison results of different methods on the KTH dataset for long time prediction experiment.

| Method | SSIM | MSE |
|---|---|---|
| Ours(B-B) | **0.882** | **80.3** |
| CM+ConvLSTM (F-F) | 0.701 | 103.4 |
| PredNet [14] | 0.656 | 108.9 |
| PredRNN++ [15] | 0.865 | 86.7 |

It should be noted that PredNet, because of the sparse nature of the prediction errors, is very fast to train (8 hours in our cluster), HPNet in frame-frame scheme took 9.3 hours while PredRNN++ and HPNet (B-B) take 10.6 hours and 11.8 hours to train respectively. Sparsifying the feedforward computation in HPNet alone reduces the training of HPNet by 13%. One might expect some additional saving if the LSTM's representations are also sparsified.

## 4.2 Evaluation of the Latent Representation in the Hierarchy

To understand how the recurrent feedback might have changed the hierarchical representation of HPNet, We trained the HPNet networks in the Block-to-Block (B-B) scheme with variable numbers of modules. First, we found that adding cortical modules tends to improve performance. Second, when we used t-SNE [50] to visualize the representation $R$ in the different modules of the networks for the last 20 dead-reckoning predicted frames of 600 testing sequences belonging to the six movements in the KTH dataset, we found that adding higher modules lead to the formation of more distinct clusters of global movement patterns in the representation units of the lower modules (Figure 4a versus Figure 4d&e; Figure 4c versus Figure 4e&h.). Better encoding of these global movement patterns in the population codes of the earlier modules manifest in the improvement of their accuracy of decoding the six movement patterns.

Table 3 compares the accuracy of decoding the six movement patterns based on the representations at four different layers (modules) of HPNet, PredNet and PredRNN++. Chance accuracy is 16%. Decoding accuracy of PredNet is close to chance because its lacking hierarchical feature representations. Decoding accuracy of PredRNN++ peaks at layer (module) 2 and 3 at 30%. For HPNet, semantic clustering and decoding accuracy improves progressively as one moves up the hierarchy, from 26% in the first layer (module) to 63% for the top module of the 4-module network. Thus, the better performance of HPNet in long range video predictions might be attributed to its ability to learn semantically meaningful hierarchical spatiotemporal feature representations and movement to movement relationships (see also [51]).

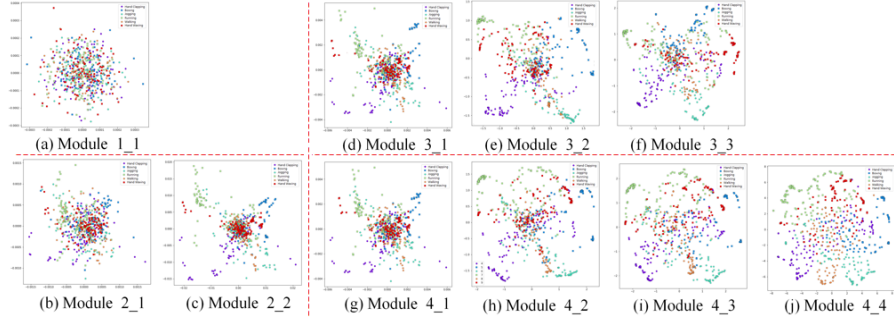

Figure 4: Visualization of R representational units of the different modules in (a) a one-module network; (b)-(c) a two-module network; (d)-(f) a three-module network; and (g)-(j) a four-module network. "Module 2_1" means Module 1 in a two-module network.

Table 3: Models' decoding results of six movement classes in the KTH dataset based on representations in the different layers of the network.

| Model with mean decoding accuracy | Layer 1 | Layer 2 | Layer 3 | Layer 4 |
|---|---|---|---|---|
| HPNet | 0.26 | 0.41 | 0.57 | 0.63 |
| PredNet | 0.19 | 0.18 | 0.16 | 0.16 |
| PredRNN++ | 0.19 | 0.30 | 0.28 | 0.18 |

### 4.3 Visual Sequence Learning in the Visual Cortex

The recurrent feedback in HPNet allows the representational (R) units in even the lower Cortical Modules to develop sensitivity to global movement and image patterns, despite these units' very localized receptive fields (Figure 4). Assuming HPNet is a plausible cortical model at least at a functional level, it predicts that neurons in the early visual areas such as V1 and V2 would exhibit sensitivity to the memory of global movement patterns.

To test this prediction, we performed a video learning neurophysiological experiment on V2 neurons in two awake behaving monkeys with Gray-Matter semi-chronic multielectrode arrays (SC32 and SC96) implanted over their V1 operculum[3]. Six experiments were carried out. Each lasted over 7 days, with daily recording sessions. In each daily session, we presented a set of 20, 800-ms long movie clips to the monkey, 20-25 times a day, so that over time, this set of movies became familiar (and predictable) to the monkey. This set is called the Predicted set or Familiar set. Every day, we also tested another set of 20 movie clips that were different daily. These sets are called the Unpredicted sets or Novel sets. Both sets of movies (through an aperture of $8^o$ in diameter over all the receptive fields of recorded neurons) were presented daily, one clip per trial, at the same location on the computer monitor relative to the red spot the monkeys fixated on during each trial. With this experimental paradigm, we can compute and compare the daily temporal responses (PSTH or peri-stimulus histogram) of all the neurons across all the movies in the Predicted set and in the Unpredicted set to monitor the development of sensitivity to memory of the familiar or predicted movies.

Since we were averaging across many neurons (over 30+ per session) with different feature preferences or tuning properties over 20 different movies, the average PSTH responses (across neurons and movies) were expected to be the same for the Predicted set and the daily Unpredicted set. Indeed, we found the averaged PSTHs for the Predicted set and the Unpredicted set to be the same for the first two or three days (Figure 5b (top row)), but they started to bifurcate at the later part of their responses in subsequent days (Figure 5b (bottom row)) with suppression for the Predicted Movies. Similar predictive suppression effects have been observed in IT neurons, and were considered as a biomarker for sequence memory. What is novel and unanticipated about our finding is that V2 neurons' receptive fields are very local, yet this "memory effect" depends on the presence of the global context of the entire movie – reducing the movie aperture from $8^o$ to $3^o$, barely larger than the

receptive fields of the individual neurons, would annihilate the predictive suppression effects. Thus, this predictive suppression effects were not due to adaptation of local receptive field features, but reflected a sensitivity to the memory of the global context of the movies or movement. Such memories are likely mediated by horizontal or recurrent feedback mechanisms or both. To better assess the evolution of the memory effect of the neuronal populations, we computed the predictive suppression index for each individual neuron as $(P - U)/(P - U)$ where $P$ and $U$ are the daily average spike counts of the neuron in the later part of responses for the Predicted set and the Unpredicted set, respectively. Figure 5a traced the development of this predictive suppression effect in one experiment, showing that the most neurons exhibit predictive suppression on the average after 3 days of exposure to the movies.

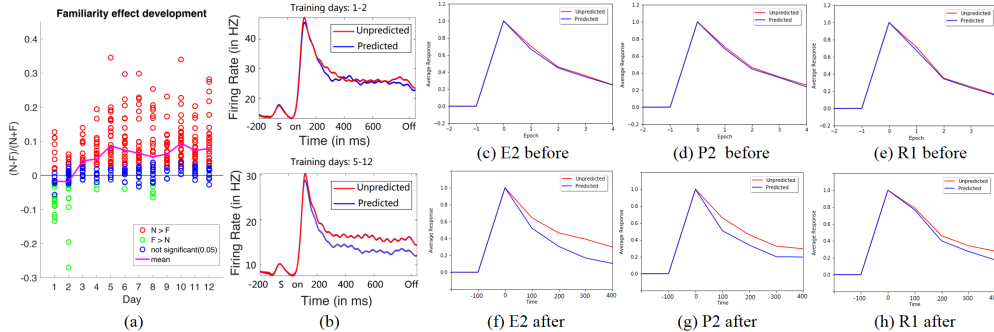

Figure 5: (a) The development of the prediction suppression effect across days in one experiment. Each dot is the prediction suppression index of a neuron. Color indicates whether the effect was significant or not (red - significant, blue - insignificant, green - significant in the opposite way) based on t-test with $p < 0.05$ as statistical significance threshold. (b) Averaged temporal responses of the V2 neurons (averaged across 3 experiments) of one monkey to Predicted set and the Unpredicted sets in the first two days (top row). Their averaged responses ( from day 5 to day 12) to the Predicted set and the Unpredicted sets, exhibiting significant prediction suppression. Module 2's normalized averaged population responses of the three types of units to the Predicted set and the Unpredicted set before (c)-(e) and after (f)-(h) training.

We performed a similar experiment on our network, pre-trained with the KTH dataset. We randomly extracted 20 sequences from the BAIR dataset [65], resized the sequence length to 40 frames and frame size to 64×64. We separated the 20 sequences into two sets – the Predicted set and the Unpredicted set. We averaged the responses to the two movie sets respectively of each type of neurons in the network ($E$ (prediction error units), $P$ (prediction units), and $R$ (representation units)) in each CM within the center 8×8 hypercolumns. Before familiarity training, the responses of each type of neurons are indeed the same for both movie sets (top row). After the network was trained with the Predicted set for 2000 epochs, prediction suppression effect can be observed in all three types of neurons (Prediction error neurons E, Representation neurons R, Prediction neurons P) in all the modules in the hierarchy, with the higher modules showing a stronger effect (see the Supplementary Material). Figure 5 (c)-(h) show the effects in CM2, corresponding roughly to V2 in the hierarchy.

It is not surprising that the prediction error neurons $E$ would decrease their responses as the network learns to predict the familiar movies better. However, it is rather interesting to find the representation neurons $R$ and the prediction neurons $P$ also exhibit prediction suppression, even though these neurons represent features rather than prediction errors. The precise reasons remain to be determined, but the fact that all neuron types in the *model* exhibited the prediction suppression effect might explain why the prediction suppression effects were commonly observed in most of the randomly sampled neurons in the visual cortex. We also performed the same experiment on PredNet and PredRNN++ and found that their corresponding R neurons would also exhibit predictive suppression effect to a certain extent, but much smaller in magnitude. Thus, we can only claim that the neurophysiological finding is consistent with this genre of self-supervised predictive learning models, though HPnet might be a better approximation. Further experiments and analysis are needed to obtain a better model approximation of the cortical mechanisms. Given PredRNN++ has no explicit feedback between LSTMs in the hierarchy, the fact that some small predictive suppression effect can be observed in

layer 2 of their R units suggests that at least part of the suppression effect is mediated by horizontal recurrent connections, which is then further enhanced by feedback.

An important question is whether the observed video prediction suppression effect in macaque might arise from the same mechanism underlying the static image familiarity suppression effects that have been long observed in the inferotemporal cortex [53,55,56] and recently in V2 [2] of macaque monkeys. The experimental paradigms are similar to the video prediction experiment we presented except that static images rather than videos were used. Similar to our video prediction experiments, after several days of exposure training, it was found that neural responses to the familiar images were significantly suppressed relative to the novel images in the later part of their responses in both IT [53,55,56] and V2 [2], with the temporal response profiles (PSTH) very similar to the ones shown in Figure 5b. It is important to note that neurons in monkey V1 and V2 have very small receptive fields, yet they show familiarity suppression effects to large object images much larger than the size of their classical receptive fields, hence there is no response difference between the averaged responses to familiar and novel when the stimuli are presented in an aperture slightly larger than the receptive fields of the neurons [2]. Together with the observation that the latency of the familiarity suppression effect was 100 ms after stimulus onset, these findings implicate the encoding of global image memories in local recurrent circuits within each visual area. We found that HPNet also exhibited the static image familiarity suppression effect when tested with static images, suggesting static image familiarity suppression effect and prediction suppression effect can both arise from the same network mechanisms. This allows us to investigate whether other deep networks trained for ImageNet image classification also exhibit familiarity suppression effects. We evaluated several supervised-learning models: standard VGG [66], and a recurrent network called deep Predictive coding networks (PCN) [67]. PCN, also inspired by predictive coding theory, used predition error minimization to derive a recurrent network architecture to be trained by backpropagation, but did not use it as a part of its objective function. Remarkably, it can achieve, with only 9 layers, ImageNet classificaiton performance comparable to a 100+-layer ResNet [68]. Nevertheless, both VGG and PCN's neurons did not exhibit the static image familiarity effect. We also experimented with a recurrent network [69] our group developed with biologically inspired top-down and horizontal recurrent connections between different stages of the VGG16. While this network yields improvement in its robustness against noises and occlusion in image classification, it also does not produce static image familiarity effect. These findings suggest that having prediction error minimization in the objective function might be an important factor for the emergence of prediction suppression or image familiarity suppression effects.

## 5   Conclusion

In this paper, we report a novel neurophysiological finding suggesting repeated exposure can induce encoding of global video memory in the early visual cortex of primates with repeated exposure. We show that a class of self-supervised prediction learning models also exhibits similar neurophysiological phenomenon. In developing the proposed hierarchical prediction network (HPNet), we found, first, sparse coding can considerably speed up computation and learning; second, processing videos by spatiotemporal blocks rather than by frame as in other models allow the LSTMs to learn relationships between movement patterns; and third, having a feature hierarchy allows explicit encoding of more and more complex movement patterns, yielding significant improvement in long-range video prediction. We found that the recurrent feedback imbued better semantic clustering of movement patterns in the early levels of hierarchical representations of HPNet, resulting in better movement pattern decoding and action recognition, even though the network was not trained to do that. To compare with representations in the visual cortex in more fine-grained details, HPNet will need to be trained with more complex and naturalistic movies. Taken together, the findings in this paper suggest the relevance of a class of predictive learning models for understanding the principles and mechanisms of learning and computation in the visual cortex.

### Acknowledgments

This work is supported by National Science Foundation (1816568). We thank Yimeng Zhang, Mert Inan, Harold Rockwell, Siming Yan, Maureen Kelly and David Pane for their technical assistance.

## Footnotes

[1] http://yann.lecun.com/exdb/mnist/

[2] http://www.nada.kth.se/cvap/actions/

[3] All experimental procedures were approved by Carnegie Mellon University's Institutional Animal Care and Use Committee, in compliance with the guidelines set forth in the United States Public Health Service Guide for the Care and Use of Laboratory Animals.

# References

[1] James L. McClelland & Bruce L. McNaughton. (1999) Complementary learning systems 1 why there are complementary learning systems in the hippocampus and neocortex : Insights from the successes and failures of connectionist models of learning and memory.

[2] Huang, G., Ramachandran, S., Lee, T. S., & Olson, C. R. (2018) Neural correlate of visual familiarity in macaque area v2. The Journal of neuroscience : the official journal of the Society for Neuroscience.

[3] Yao, H., Shi, L., Han, F., Gao, H., & Dan, Y. (2007) Rapid learning in cortical coding of visual scenes. Nature Neuroscience, 10:772–778.

[4] Han, F., Caporale, N., & Dan, Y. (2008) Reverberation of recent visual experience in spontaneous cortical waves. Neuron, 60:321–327.

[5] Xu, S., Jiang, W., Poo, M.-M., & Dan, Y. (2012) Activity recall in visual cortical ensemble. In Nature Neuroscience.

[6] Cooke, S. F. & Bear, M. F. (2014) How the mechanisms of long-term synaptic potentiation and depression serve experience-dependent plasticity in primary visual cortex. Philosophical transactions of the Royal Society of London. Series B, Biological sciences, 369 1633:20130284.

[7] Cooke, S. F. & Bear, M. F. (2015) Visual recognition memory: a view from v1. Current opinion in neurobiology, 35: 57–65.

[8] Elman, J. L. (1990) Finding structure in time. Cognitive Science, 14:179–211.

[9] Mathieu, M., Couprie, C., & LeCun, Y. (2015) Deep multiscale video prediction beyond mean square error. CoRR, abs/1511.05440.

[10] Villegas, R., Yang, J., Zou, Y., Sohn, S., Lin, X., & Lee, H. (2017) Learning to generate long-term future via hierarchical prediction. In ICML.

[11] Srivastava, N., Mansimov, E., & Salakhutdinov, R. (2015) Unsupervised learning of video representations using lstms. In ICML.

[12] O'Reilly, R. C., Wyatte, D., & Rohrlich, J. (2014) Learning through time in the thalamocortical loops.

[13] Lee, T. S. (2015) The visual system's internal models of the world. Proceedings of the IEEE, 103:1359–1378.

[14] Lotter, W., Kreiman, G., & Cox, D. D. (2016) Deep predictive coding networks for video prediction and unsupervised learning. CoRR, abs/1605.08104.

[15] Wang, Y., Gao, Z., Long, M., Wang, J., & Yu, P. S. (2018) Predrnn++: Towards a resolution of the deep-in-time dilemma in spatiotemporal predictive learning. In ICML.

[16] Mumford, D. (1991) On the computational architecture of the neocortex. Biological Cybernetics, 65:135–145.

[17] Rao, R. P. N. & Ballard, D. H. (1999) Predictive coding in the visual cortex: a functional interpretation of some extraclassical receptive-field effects. Nature neuroscience, 2 1: 79–87.

[18] Dijkstra, N., Zeidman, P., Ondobaka, S., van Gerven, M. A. J., & Friston, K. J. (2017) Distinct top-down and bottom-up brain connectivity during visual perception and imagery. In Scientific Reports.

[19] Friston, K. J. (2018) Does predictive coding have a future? Nature Neuroscience, 21:1019–1021.

[20] Choy, C. B., Xu, D., Gwak, J., Chen, K., & Savarese, S. (2016) 3d-r2n2: A unified approach for single and multi-view 3d object reconstruction. In ECCV.

[21] Mumford, D. (1992) On the computational architecture of the neocortex. ii. the role of cortico-cortical loops. Biological cybernetics, 66 3:241–51.

[22] McClelland, J. L. & Rumelhart, D. E. (1985) Distributed memory and the representation of general and specific information. Journal of experimental psychology. General, 114 2:159– 97.

[23] Grossberg, S. (1987) Competitive learning: From interactive activation to adaptive resonance. Cognitive Science, 11:23–63.

[24] Palm, R. B. (2012) Prediction as a candidate for learning deep hierarchical models of data.

[25] Goroshin, R., Mathieu, M., & LeCun, Y. 92015 Learning to linearize under uncertainty. In NIPS.

[26] Patraucean, V., Handa, A., & Cipolla, R. (2015) Spatio-temporal video autoencoder with differentiable memory. CoRR, abs/1511.06309.

[27] Vondrick, C., Pirsiavash, H., & Torralba, A. (2016) Generating videos with scene dynamics. In NIPS.

[28] Kalchbrenner, N., van den Oord, A., Simonyan, K., Danihelka, I., Vinyals, O., Graves, A., a& Kavukcuoglu, K. (2017) Video pixel networks. In ICML.

[29] Xu, Z., Wang, Y., Long, M., & Wang, J. (2018) Predcnn: Predictive learning with cascade convolutions. In IJCAI.

[30] Oh, J., Guo, X., Lee, H., Lewis, R. L., & Singh, S. P. (2015) Action-conditional video prediction using deep networks in atari games. In NIPS.

[31] Lee, A. X., Zhang, R., Ebert, F., Abbeel, P., Finn, C., & Levine, S. (2018) Stochastic adversarial video prediction. CoRR, abs/1804.01523.

[32] Wichers, N., Villegas, R., Erhan, D., & Lee, H. (2018) Hierarchical long-term video prediction without supervision. In ICML.

[33] Finn, C., Goodfellow, I. J., & Levine, S. (2016) Unsupervised learning for physical interaction through video prediction. In NIPS.

[34] Wang, Y., Long, M., Wang, J., Gao, Z., & Yu, P. S. (2017) Predrnn: Recurrent neural networks for predictive learning using spatiotemporal lstms. In NIPS.

[35] Wang, Y., Jiang, L., Yang, M.-H., Li, L.-J., Long, M., & Fei-Fei, L. (2019) Eidetic 3d lstm: A model for video prediction and beyond. In ICLR.

[36] Babaeizadeh, M., Finn, C., Erhan, D., Campbell, R. H., & Levine, S. (2017) Stochastic variational video prediction. CoRR, abs/1710.11252.

[37] Denton, E. L. & Fergus, R. (2018) Stochastic video generation with a learned prior. In ICML.

[38] Ullman, S. (1995) Sequential seeking and counter streams: a computational model for bidirectional flow in the visual cortex. Cerebral Cortex, 5:1:1–1.

[39] Lee, T. S. & Mumford, D. (2003) Hierarchical bayesian inference in the visual cortex. Journal of the Optical Society of America. A, Optics, image science, and vision, 20 7:1434– 48.

[40] Dayan, P., Hinton, G. E., Neal, R. M., & Zemel, R. S. (1995) The helmholtz machine. Neural Computation, 7:889–904.

[41] Kersten, D. J. & Yuille, A. L. (2003) Bayesian models of object perception. Current opinion in neurobiology, 13 2:150–8.

[42] Nayebi, A., Bear, D., Kubilius, J., Kar, K., Ganguli, S., Sussillo, D., DiCarlo, J. J., & Yamins, D. L. K. (2018) Taskdriven convolutional recurrent models of the visual system. CoRR, abs/1807.00053.

[43] Wen, H., Han, K., Shi, J., Zhang, Y., Culurciello, E., & Liu, Z. (2018) Deep predictive coding network for object recognition. In ICML.

[44] Pan, B., Lin, W., Fang, X., Huang, C., Zhou, B., & Lu, C. (2018) Recurrent residual module for fast inference in videos. CoRR, abs/1802.09723.

[45] Liu, X., Pool, J., Han, S., & Dally, W. J. (2017) Efficient sparse-winograd convolutional neural networks. CoRR, abs/1802.06367.

[46] Dave, A., Russakovsky, O., & Ramanan, D. (2017) Predictivecorrective networks for action detection. 2017 IEEE Conference on Computer Vision and Pattern Recognition (CVPR), pp. 2067–2076.

[47] Shi, X., Chen, Z., Wang, H., Yeung, D.-Y., Wong, W.-K., & chun Woo, W. (2015) Convolutional lstm network: A machine learning approach for precipitation nowcasting. In NIPS.

[48] Wang, Z., Bovik, A. C., Sheikh, H. R., & Simoncelli, E. P. (2004) Image quality assessment: from error visibility to structural similarity. IEEE Transactions on Image Processing, 13:600–612.

[49] Schuldt, C., Laptev, I., & Caputo, B. (2004) Recognizing human  International Conference on Pattern Recognition, 2004. ICPR 2004., 3:32–36 Vol.3.

[50] van der Maaten, L. & Hinton, G. E. (2008) Visualizing data using t-sne.

[51] Kheradpisheh, S. R., Ganjtabesh, M., Thorpe, S. J., & Masquelier, T. (2018) Stdp-based spiking deep convolutional neural networks for object recognition. Neural networks : the official journal of the International Neural Network Society, 99:56–67.

[52] Meyer, T. & Olson, C. R. (2011) Statistical learning of visual transitions in monkey inferotemporal cortex. Proceedings of the National Academy of Sciences of the United States of America, 108 48:19401–6.

[53] Meyer, T., Walker, C., Cho, R. Y., & Olson, C. R. (2014) Image Familiarization sharpens response dynamics of neurons in inferotemporal cortex. Nature Neuroscience, 17:1388– 1394.

[54] Ramachandran, S., Meyer, T., & Olson, C. R. (2017) Prediction suppression and surprise enhancement in monkey inferotemporal cortex. Journal of neurophysiology, 118 1:374–382.

[55] Freedman, D. J. & Assad, J. A. (2006) Experience-dependent representation of visual categories in parietal cortex. Nature, 443:85–88.

[56] Mruczek, R. E. B. & Sheinberg, D. L. (2007) Context familiarity enhances target processing by inferior temporal cortex neurons. The Journal of neuroscience : the official journal of the Society for Neuroscience, 27 32:8533–45.

[58] Yao, H., Shi, L., Han, F., Gao, H., & Dan, Y. (2007) Rapid learning in cortical coding of visual scenes. Nature Neuroscience, 10:772–778.

[59] Han, F., Caporale, N., & Dan, Y. (2008) Reverberation of recentvisual experience in spontaneous cortical waves. Neuron, 60:321–327, 2008.

[60] Xu, S., Jiang, W., Poo, M.-M., & Dan, Y. (2012) Activity recall in visual cortical ensemble. In Nature Neuroscience.

[61] Hawkins, J. and George, D. (2006) Hierarchical temporal memory concepts, theory, and terminology.

[62] Ullman, S. (1995) Sequential seeking and counter streams: a computational model for bidirectional flow in the visual cortex. Cerebral Cortex, 5:1:1–1.

[63] Lotter W., Kreiman, G., Cox, D. (2018) A neural network trained to predict future video frames mimics critical properties of biological neuronal responses and perception. In NIPS.

[64] Lee, T. S. and Mumford, D.(2003) Hierarchical bayesian inference in the visual cortex. Journal of the Optical Society of America. A, Optics, image science, and vision, 20 7:1434–48.

[65] Ebert, F., Finn, C., Lee. A., and Levine, S. (2017) Self-supervised visual planning with temporal skip connections. In Conference on Robot Learning (CoRL).

[66] Simonyan, K., Zisserman, A. (2014) Very deep convolutional networks for large-scale image recognition. arXiv: 1409.1556.

[67] Han,K., Wen, H., Fu, D., Culurciello E., and Liu Z (2018) Deep predictive coding network with local recurrent processing for object recognition. NIPS 18, Proceedings of the 32nd International Conference on Neural Information Processing Systems 9221-9233. Montreal Canada.

[68] He, K., Zhang, X., Ren, S., and Sun, J. (2016) Deep residual learning for image recognition. In Proceedings of the IEEE conference on computer vision and pattern recognition,pages 770–778, 2016.

[69] Yan, S., Fang, X, Xiao, B., Rockwell, H., Zhang, Y., Lee, T.S. (2019) Recurrent feedback improves feedforward representations in deep neural networks. arXiv.

